# Query-Aware MCMC

**Michael Wick**
Department of Computer Science
University of Massachusetts
Amherst, MA
mwick@cs.umass.edu

**Andrew McCallum**
Department of Computer Science
University of Massachusetts
Amherst, MA
mccallum@cs.umass.edu

## Abstract

Traditional approaches to probabilistic inference such as loopy belief propagation and Gibbs sampling typically compute marginals for *all* the unobserved variables in a graphical model. However, in many real-world applications the user's interests are focused on a subset of the variables, specified by a query. In this case it would be wasteful to uniformly sample, say, one million variables when the query concerns only ten. In this paper we propose a query-specific approach to MCMC that accounts for the query variables and their generalized mutual information with neighboring variables in order to achieve higher computational efficiency. Surprisingly there has been almost no previous work on query-aware MCMC. We demonstrate the success of our approach with positive experimental results on a wide range of graphical models.

## 1 Introduction

Graphical models are useful for representing relationships between large numbers of random variables in probabilistic models spanning a wide range of applications, including information extraction and data integration. Exact inference in these models is often computationally intractable due to the dense dependency structures required in many real world problems, thus there exists a large body of work on both variational and sampling approximations to inference that help manage large treewidth. More recently, however, inference has become difficult for a different reason: large data. The proliferation of interconnected data and the desire to model it has given rise to graphical models with millions or even billions of random variables. Unfortunately, there has been little research devoted to approximate inference in graphical models that are large in terms of their number of variables. Other than acquiring more machines and parallelizing inference [1, 2], there have been few options for coping with this problem.

Fortunately, many inference needs are instigated by queries issued by users interested in particular random variables. These real-world queries tend to be grounded (i.e., focused on specific data cases). For example, a funding agency might be interested in the expected impact that funding a *particular* research group has on a *certain* scientific topic. In these situations not all variables are of equal relevance to the user's query; some variables become observed given the query, others become statistically independent given the query, and the remaining variables are typically marginalized. Thus, a user-generated query provides a tremendous amount of information that can be exploited by an intelligent inference procedure. Unfortunately, traditional approaches to inference such as loopy belief propagation (BP) and Gibbs sampling are query agnostic in the sense that they fail to take advantage of this knowledge and treat each variable as equally relevant. Surprisingly, there has been little research on query specific inference and the only existing approaches focus on loopy BP [3, 4].

In this paper we propose a query-aware approach to Markov chain Monte Carlo (QAM) that exploits the dependency structure of the graph and the query to achieve faster convergence to the answer. Our method selects variables for sampling in proportion to their influence on the query variables. We

determine this influence using a computationally tractable generalization of mutual information between the query variables and each variable in the graph. Because our query-specific approach to inference is based on MCMC, we can provide arbitrarily close approximations to the query answer while also scaling to graphs whose structure and unrolled factor density would ordinarily preclude both exact and belief propagation inference methods. This is essential for the method to be deployable in real-world probabilistic databases where even a seemingly innocuous relational algebra query over a simple fully independent structure can produce an inference problem that is #P-hard [5]. We demonstrate dramatic improvements over traditional Markov chain Monte Carlo sampling methods across a wide array of models of diverse structure.

## 2 Background

### 2.1 Graphical Models

Graphical models are a flexible framework for capturing statistical relationships between random variables. A factor graph $\mathcal{G} := \langle \mathbf{x}, \boldsymbol{\psi} \rangle$ is a bipartite graph consisting of $n$ random variables $\mathbf{x} = \{x_i\}_1^n$ and $m$ factors $\boldsymbol{\psi} = \{\psi_i\}_1^m$. Each variable $x_i$ has a domain $\mathcal{X}_i$, and we notate the entire domain space of the random variables ($\mathbf{x}$) as $\mathcal{X}$ with associated $\sigma$-algebra $\Omega$. Intuitively, a factor $\psi_i$ is a function that maps a subset of random variable values $v^i \in \mathcal{X}^i$ to a non-negative real-valued number, thus capturing the compatibility of an assignment to those variables. The factor graph then expresses a probability measure over $(\mathcal{X}, \Omega)$, the probability of a particular event $\boldsymbol{\omega} \in \Omega$ is given as

$$\pi(\omega) = \frac{1}{Z} \sum_{\mathbf{v} \in \boldsymbol{\omega}} \prod_{i=1}^{m} \psi_i(v^i), \quad Z = \sum_{\mathbf{v} \in \mathcal{X}} \pi(\mathbf{v}). \tag{1}$$

We will assume that $\Omega$ is defined so that marginalization of any subset of the variables is well defined; this is important in the sequel.

### 2.2 Queries on Graphical Models

Informally, a query on a graphical model is a request for some quantity of interest that the graphical model is capable of providing. That is, a query is a function mapping the graphical model to an answer set. Inference is required to recover these quantities and produce an answer to the query. While in the general case, a query may contain arbitrary functions over the support of a graphical model, for this work we consider queries of the marginal form. That is a query $Q$ consists of three parts $Q = \langle \mathbf{x}_q, \mathbf{x}_l, \mathbf{x}_e \rangle$. Where $\mathbf{x}_q$ is the set of query variables whose marginal distributions (or MAP configuration) are the answer to the query, $\mathbf{x}_e$ is a set of evidence variables whose values are observed, and $\mathbf{x}_l$ is the set of latent variables over which one typically marginalizes to obtain the statistically sound answer. Note that this class of queries is remarkably general and includes queries that require expectations over arbitrary functions. We can see this because a function over the graphical model (or a subset of the graphical model) is itself a random variable, and can therefore be included in $\mathbf{x}_q$.[1] More precisely, a query over a graphical model is:

$$Q(\mathbf{x}_q, \mathbf{x}_l, \mathbf{x}_e, \pi) = \pi(\mathbf{x}_q | \mathbf{x}_e = \mathbf{v}_e) = \sum_{\mathbf{v}_l} \pi(\mathbf{x}_q, \mathbf{x}_l | \mathbf{x}_e = \mathbf{v}_e) \tag{2}$$

we assume that $\Omega$ is well defined with respect to marginalization over arbitrary subsets of variables.

### 2.3 Markov Chain Monte Carlo

Markov chain Monte Carlo (MCMC) is an important inference method for graphical models where computing the normalization constant $Z$ is intractable. In particular, for many MCMC schemes such as Gibbs sampling and more generally Metropolis-Hastings, $Z$ cancels out of the computation for generating a single sample. MCMC has been successfully used in a wide variety of applications including information extraction [8], data integration [9], and machine vision [10]. For simplicity, in this work, we consider Markov chains over discrete state spaces. However, many of the results

presented in this paper may be extended to arbitrary state spaces using more general statements with measure theoretic definitions.

Markov chain Monte Carlo produces a sequence of states $\{s_i\}_1^\infty$ in a state space $S$ according to a transition kernel $K : S \times S \to \mathbb{R}_+$, which in the discrete case is a stochastic matrix: for all $s \in S$ $K(s, \cdot)$ is a valid probability measure and for all $s \in S$ $K(\cdot, s)$ is a measurable function. Since we are concerned with MCMC for inference in graphical models, we will from now on let $S := \mathcal{X}$, and use $\mathcal{X}$ instead. Under certain conditions the Markov chain is said to be ergodic, then the chain exhibits two types of convergence. The first is of practical interest: a law of large numbers convergence

$$\lim_{t \to \infty} \frac{1}{t} \sum f(s_t) = \int_{s \in \mathcal{X}} f(s)\pi(s)ds \tag{3}$$

where the $s_t$ are empirical samples from the chain.

The second type of convergence is to the distribution $\pi$. At each time step, the Markov chain is in a time-specific distribution over the state space (encoding the probability of being in a particular state at time $t$). For example, given an initial distribution $\pi_0$ over the state space, the probability of being in a next state $s'$ is the probability of all paths beginning in starting states $s$ with probabilities $\pi_0(s)$ and transitioning to $s'$ with probabilities $K(s, s')$. Thus the time-specific ($t = 1$) distribution over all states is given by $\pi^{(1)} = \pi_0 K$; more generally, the distribution at time $t$ is given by $\pi^{(t)} = \pi_0 K^t$. Under certain conditions and regardless of the initial distribution, the Markov chain will converge to the stationary (invariant) distribution $\pi$. A sufficient (but not necessary) condition for this is to require that the Markov transition kernel obey detailed balance:

$$\pi(x)K(x, x') = \pi(x')K(x', x) \quad \forall x, x' \in \mathcal{X} \tag{4}$$

Convergence of the chain is established when repeated applications of the transition kernel maintain the invariant distribution $\pi = \pi K$, and convergence is traditionally quantified using the total variation norm:

$$\|\pi^{(t)} - \pi\|_{\text{tv}} := \sup_{\mathcal{A} \in \Omega} |\pi^{(t)}(\mathcal{A}) - \pi(\mathcal{A})| = \frac{1}{2} \sum_{\mathbf{x} \in \mathcal{X}} |\pi^{(t)}(\mathbf{x}) - \pi(\mathbf{x})| \tag{5}$$

The rate at which a Markov chain converges to the stationary distribution is proportional to the spectral gap of the transition kernel, and so there exists a large body of literature proving bounds on the second eigenvalues.

## 2.4 MCMC Inference in Graphical Models

MCMC is used for inference in graphical models by constructing a Markov chain with invariant distribution $\pi$ (given by the graphical model). One particularly successful approach is the Metropolis Hastings (MH) algorithm. The idea is to devise a proposal distribution $T : \mathcal{X} \times \mathcal{X} \to [0, 1]$ for which it is always tractable to sample a next state $s'$ given a current state $s$. Then, the proposed state $s'$ is accepted with probability function $A$

$$A(s, s') = \min\left(1, \frac{\pi(s')T(s, s')}{\pi(s)T(s', s)}\right) \tag{6}$$

The resulting transition kernel $K_{\text{MH}}$ is given by

$$K_{MH}(s, s') = \begin{cases} T(s, s') & \text{if } A(s, s') \geqslant 1, s \neq s' \\ T(s, s')A(s, s') & \text{if } A(s, s') < 1 \\ T(s, s') + \sum_{r:A(s,r)<1} K(s, r)(1 - A(s, r)) & \text{if } s = s' \end{cases} \tag{7}$$

Further, observe that in the computation of A, the partition function $Z$ cancels, as do factors outside the Markov blanket of the variables that have changed. As a result, generating samples from graphical models with Metropolis-Hastings is usually inexpensive.

## 3 Query Specific MCMC

Given a query $Q = \langle \mathbf{x}_q, \mathbf{x}_l, \mathbf{x}_e \rangle$, and a probability distribution $\pi$ encoded by a graphical model $\mathcal{G}$ with factors $\psi$ and random variables $\mathbf{x}$, the problem of query specific inference is to return the highest fidelity answer to $Q$ given a possible time budget. We can put more precision on this statement by defining "highest fidelity" as closest to the truth in total variation distance.

Our approach for query specific inference is based on the Metropolis Hastings algorithm described in Section 2.4. A simple yet generic case of the Metropolis Hastings proposal distribution $T$ (that has been quite successful in practice) employs the following steps:

1: Beginning in a current state $s$, select a random variable $x_i \in \mathbf{x}$ from a probability distribution $p$ over the indices of the variables $(1, 2, \cdots, n)$.
2: Sample a new value for $x_i$ according to some distribution $q(\mathcal{X}_i)$ over that variable's domain, leave all other variables unchanged and return the new state $s'$.

In brief, this strategy arrives at a new state $s'$ from a current state $s$ by simply updating the value of one variable at a time. In traditional MCMC inference, where the marginal distributions of all variables are of equal interest, the variables are usually sampled in a deterministic order, or selected uniformly at random; that is, $p(i) = \frac{1}{n}$ induces a uniform distribution over the integers $1, 2, \cdots, n$.

However, given a query $Q$, it is reasonable to choose a $p$ that more frequently selects the query variables for sampling. Clearly, the query variable marginals depend on the remaining latent variables, so we must tradeoff sampling between query and non-query variables. A key observation is that not all latent variables influence the query variables equally. A fundamental question raised and addressed in this paper is: *how do we pick a variable selection distribution $p$ for a query $Q$ to obtain the highest fidelity answer under a finite time budget.* We propose to select variables based on their *influence* on the query variable according to the graphical model.

We will now formalize a broad definition of influence by generalizing mutual information. The mutual information $I(x, y) = \pi(x, y) \log(\frac{\pi(x,y)}{\pi(x)\pi(y)})$ between two random variables measures the strength of their dependence. It is easy to check that this quantity is the KL divergence between the joint distribution of the variables and the product of the marginals: $I(x, y) = KL(\pi(x, y) \| \pi(x)\pi(y))$. In this sense, mutual information measures dependence as a "distance" between the full joint distribution and its independent approximation. Clearly, if $x$ and $y$ are independent then this distance is zero and so is their mutual information. We produce a generalization of mutual information which we term the *influence* by substituting an arbitrary divergence function $f$ in place of the KL divergence.

**Definition 1** (*Influence*). *Let $x$ and $y$ be two random variables with marginal distributions $\pi(x, y), \pi(x), \pi(y)$. Let $f(\pi_1(\cdot), \pi_2(\cdot)) \mapsto r, r \in \mathbb{R}_+$ be a non-negative real-valued divergence between probability distributions. The influence $\iota(x, y)$ between $x$ and $y$ is*

$$\iota(x, y) := f(\pi(x, y), \pi(x)\pi(y)) \tag{8}$$

If we let $f$ be the KL divergence then $\iota$ becomes the mutual information; however, because MCMC convergence is more commonly assessed with total variation norm, we define an influence metric based on this choice for $f$. In particular we define $\iota_{\text{tv}}(x, y) := \|\pi(x, y) - \pi(x)\pi(y)\|_{\text{tv}}$.

As we will now show, the total variation influence (between the query variable and the latent variables) has the important property that it is exactly the error incurred from ignoring a single latent variable when sampling values for $x_q$. For example, suppose we design an approximate query specific sampler that saves computational resources by ignoring a particular random variable $x_l$. Then, the variable $x_l$ will remain at its burned-in value $x_l = v_l$ for the duration of query specific sampling. As a result, the chain will converge to the invariant distribution $\pi(\cdot | x_l = v_l)$. If we use this conditional distribution to approximate the marginal, then the expected error we incur is exactly the influence score under total variation distance.

**Proposition 1.** *If $p(i) = \mathbb{1}(i \neq l)\frac{1}{n-1}$ induces an MH kernel that neglects variable $x_l$, then the expected total variation error $\xi_{tv}$ of the resulting MH sampling procedure under the model is the total variation influence $\iota_{tv}$.*

**Proof:** The resulting chain has stationary distribution $\pi(x_q|x_l = v_l)$. The expected error is:

$$
\begin{aligned}
\mathbb{E}_\pi[\xi_{\text{tv}}] &= \sum_{v_l \in \mathcal{X}_l} \pi(x_l{=}v_l) \| \pi(x_q|x_l{=}v_l) - \pi^{(t)}(x_q) \|_{\text{tv}} \\
&= \sum_{v_l \in \mathcal{X}_l} \pi(x_l{=}v_l) \frac{1}{2} \sum_{v_q \in \mathcal{X}_q} \left| \pi(x_q|x_l{=}v_l) - \pi^{(t)}(x_q) \right| \\
&= \frac{1}{2} \sum_{v_l \in \mathcal{X}_l} \sum_{v_q \in \mathcal{X}_q} \left| \pi(x_q|x_l{=}v_l)\pi(x_l{=}v_l) - \pi^{(t)}(x_q)\pi(x_l{=}v_l) \right| \\
&= \frac{1}{2} \sum_{v_l \in \mathcal{X}_l} \sum_{v_q \in \mathcal{X}_q} \left| \pi(x_q, x_l) - \pi^{(t)}(x_q)\pi(x_l) \right| = \iota_{\text{tv}}(x_q, x_l)
\end{aligned}
$$

$\square$

This demonstrates that the expected cost of not sampling a variable is exactly that variable's influence on the query variable. We are now justified in selecting variables proportional to their influence to reduce the error they assert on the query marginal. For example, if a variable's influence score is zero this also means that there is no cost incurred from neglecting that variable (if a query renders variables statistically independent of the query variable then these variables will be correctly ignored under the influence based sampling procedure).

Note, however, that computing either $\iota_{\text{tv}}$ or the mutual information is as difficult as inference itself. Thus, we define a computationally efficient variant of influence that we term the *influence trail score*. The idea is to approximate the true influence as a product of factors along an active trail in the graph.

**Definition 2** (Influence Trail Score). *Let $\rho = (x_0, x_1, \cdots, x_r)$ be an active trail between the query variable $x_q$ and $x_i$ where $x_0 = x_q$ and $x_r = x_i$. Let $\phi(x_i, x_j)$ be the approximate joint distribution between $x_i$ and $x_j$ according only to the mutual factors in their scopes. Let $\phi(x_i) = \sum_{x_j} \phi(x_i, x_j)$ be a marginal distribution. The influence trail score with respect to an active trail $\rho$ is*

$$
\tau_\rho(x_q, x_i) := \prod_{i=1}^{r-1} f(\phi_i(x_i, x_{i+1}), \phi_i(x_i)\phi_i(x_{i+1})) \tag{9}
$$

The influence trail score is efficient to compute because all factors and variables outside the mutual scopes of each variable pair are ignored. In the experimental results we evaluate both the influence and the influence trail and find that they perform similarly and outperform competing graph-based heuristics for determining $p$.

While in general it is difficult to uniformly state that one choice of $p$ converges faster than another for all models and queries, we present the following analysis showing that even an approximate query aware sampler can exhibit faster finite time convergence progress than an exact sampler. Let $K$ be an exact MCMC kernel that converges to the correct stationary distribution and let $L$ be an approximate kernel that exclusively samples the query variable and thus converges to the conditional distribution of the query variable. We now assume an ergodic scheme for the two samplers where the convergence rates are geometrically bounded from above and below by constants $\gamma_l$ and $\gamma_k$:

$$
\| \pi_0 L^t - \pi_K \|_{\text{tv}} = \Theta(\gamma_l^t) \tag{10}
$$

$$
\| \pi_0 K^t - \pi_K \|_{\text{tv}} = \Theta(\gamma_k^t) \tag{11}
$$

Because $L$ only samples the query variable, the dimensionality of $L$'s state space is much smaller than $K$'s state space, and we will assume that $L$ converges more quickly to its own invariant distribution, that is, $\gamma_l \ll \gamma_k$. Extrapolating Proposition 1, we know that the error incurred from neglecting to sample the latent variables is the influence $\iota_{\text{tv}}$ between the joint distribution of the latent variables and the query variable. Observe that $L$ is simultaneously making progress towards two distributions: its own invariant distribution and the invariant distribution of $K$ plus an error term. If the error term $\iota_{\text{tv}}$ is sufficiently small then we can write the following inequality:

$$
\gamma_l^t + \iota_{\text{tv}} \le \gamma_k^t \tag{12}
$$

We want this inequality to hold for as many time steps as possible. The amount of time that $L$ (the query only kernel) is closer to $K$'s stationary distribution $\pi_k$ can be determined by solving for $t$,

yielding the fixed point iteration:

$$t = \frac{\log\left(\gamma_l^t + \iota_{\text{tv}}\right)}{\log \gamma_k} \tag{13}$$

The one-step approximation yields a non-trivial, but conservative bound: $t \geq \frac{\log(\gamma_l + \iota_{\text{tv}})}{\log \gamma_k}$. Thus, for a sufficiently small error, $t$ can be positive. This implies that the strategy of exclusively sampling the query variables can achieve faster short-term convergence to the correct invariant distribution even though asymptotic convergence is to the incorrect invariant distribution. Indeed, we observe this phenomena experimentally in Section 5.

## 4 Related Work

Despite the prevalence of probabilistic queries, the machine learning and statistics communities have devoted little attention to the problem of query-specific inference. The only existing papers of which we are aware both build on loopy belief propagation [3, 4]; however, for many inference problems, MCMC is a preferred alternative to LPB because it is (1) able to obtain arbitrarily close approximations to the true marginals and (2) is better able to scale to models with large or real-valued variable domains that are necessary for state-of-the-art results in data integration [9], information extraction [8], and deep vision tasks with many latent layers [11].

To the best of our knowledge, this paper is one of the first to propose a query-aware sampling strategy for MCMC in either the machine learning or statistics community. The decayed MCMC algorithm for filtering [12] can be thought of as a special case of our method where the model is a linear chain, and the query is for the last variable in the sequence. That paper proves a finite mixing time bounds on infinitely long sequences. In contrast we are interested in arbitrarily shaped graphs and in the practical consideration of large finite models. MCMC has also recently been deployed in probabilistic databases [13] where it is possible to incorporate the deterministic constraints of a relational algebra query directly into a Metropolis-Hastings proposal distribution to obtain quicker answers [14, 15].

A related idea from statistics is data augmentation (or auxiliary variable) approaches to sampling where latent variables are artificially introduced into the model to improve convergence of the original variables (e.g., Swendsen-Wang [16] and slice sampling [17]). In this setting, we see QAM as a way of determining a more sophisticated variable selection strategy that can balance sampling efforts between the original and auxiliary variables.

## 5 Experiments

In this section we demonstrate the effectiveness and broad applicability of query aware MCMC (QAM) by demonstrating superior convergence rates to the query marginals across a diverse range of graphical models that vary widely in structure. In our experiments, we generate a wide range of graphical models and evaluate the convergence of each chain exactly, avoiding noisy empirical sampling error by performing exact computations with full transition kernels.

We evaluate the following query-aware samplers:

1. Polynomial graph distance 1 (*QAM-Poly1*): $p(x_i) \propto d(x_q, x_i)^{-N}$, where $d$ is shortest path;
2. Influence - Exact mutual information (*QAM-MI*): $p(x_i) \propto I(x_q, x_i)$;
3. Influence - total variation distance (*QAM-TV*): $p(x_i) \propto \iota_{\text{tv}}(x_q, x_i)$;
4. Influence trail score - total variation (*QAM-TV*): $p(x_i)$ set according to Equation 9;

and two baseline samplers

7. Traditional Metropolis-Hastings (*Uniform*): $p(x_i) \propto 1$;
8. Query-only Metropolis-Hastings (*qo*): $p(x_i) = \mathbb{1}(x_q = x_i)$;

on six different graphical models with varying parameters generated from a Beta(2,2) distribution (this ensures an interesting dynamic range over the event space).

1. **Independent** - each variable is statistically independent
2. **Linear chain** - a linear-chain CRF (used in NLP and information extraction)
3. **Hoop** - same as linear chain plus additional factor to close the loop
4. **Grid** - or Ising model, used in statistical physics and vision
5. **Fully connected PW** - each pair of variables has a pairwise factor
6. **Fully connected** - every variable is connected through a single factor

Mirroring many real-world conditional random field applications, the non-unary factors (connecting more than one variable) are generated from the same factor-template and thus share the same parameters (each generated from log(Beta(2,2))). Each variable has a corresponding observation factor whose parameters are not shared and randomly set according to log(Beta2,2)/2.

For our experiments we randomly generate ten parameter settings for each of the six model types and measure convergence of the six chains to the the single-variable marginal query $\pi(x_q)$ for each variable in each of the sixty realized models. Convergence is measured using the total variation norm: $\|\pi(x_q) - \pi(x_q)^{(t)}\|_{\text{tv}}$. In this set of experiments we do not wish to introduce empirical sampling error so we generate models with nine-variables per graph enabling us to (1) exactly compute the answer to the marginal query, (2) fully construct the $2^n \times 2^n$ transition matrices, and (3) algebraically compute the time $t$ distributions for each chain $\pi^{(t)} = \pi_0 K_{\text{MH}}^t$ given an initial uniform distribution $\pi_0(\mathbf{x}) = 2^{-9}$.

We display marginal convergence results in Figure 1. Generally, all the query specific sampling chains converge more quickly than the uniform baseline in the early iterations across every model. It is interesting to compare the convergence rates of the various QAM approaches at different time stages. The query-only and mutual information chain exhibit the most rapid convergence in the early stages of learning, with the query-only chain converging to an incorrect distribution, and the mutual information chain slowly converging during the later time stages. While QAM-TV exhibits similar convergence patterns to the polynomial chains, QAM-TV slightly outperforms them in the more connected models (grid and fully-connected-pw). Finally, notice that the influence-trail variant of total variation influence converges at a similar rate to the actual total variation influence, and in some cases converges more quickly (e.g., in the grid and the latter stages of the full pairwise model).

In the next experiment, we demonstrate how the size of the graphical model affects convergence of the various chains. In particular, we plot the convergence of all chains on six different hoop-structured models containing three, four, six, eight, ten, and twelve variables (Figure 2). Again, the results are averaged over ten randomly generated graphs, but this time we plot the advantage over the uniform kernel. That is we measure the difference in convergence rates $\|\pi - \pi_0 K_{\text{Unif}}^t\|_{\text{tv}} - \|\pi - \pi_0 K_{\text{QAM}}^t\|_{\text{tv}}$ so that points above the line $x = 0$ mean the QAM is closer to the answer than the uniform baseline and points below the line mean the QAM is further from the answer. As expected, increasing the number of variables in the graph increases the opportunities for query specific sampling and thus increases QAM's advantage over traditional MCMC.

## 6   Conclusion

In this paper we presented a query-aware approach to MCMC, motivated by the need to answer queries over large scale graphical models. We found that the query-aware sampling methods outperform the traditional Metropolis Hastings sampler across all models in the early time steps. Further, as the number of variables in the models increase, the query aware samplers not only outperform the baseline for longer periods of time, but also exhibit more dramatic convergence rate improvements. Thus, query specific sampling is a promising approach for approximately answering queries on real-world probabilistic databases (and relational models) that contain billions of variables. Successfully deploying QAM in this setting will require algorithms for efficiently constructing and sampling the variable selection distribution. An exciting area of future work is to combine query specific sampling with adaptive MCMC techniques allowing the kernel to evolve in response to the underlying distribution. Further, more rapid convergence could be obtained by mixing the kernels in a way that combines the strength of each: some kernels converge quickly in the early stages of sampling while other converge more quickly in the later stages, thus together they could provide a very powerful query specific inference tool. There has been little theoretical work on analyzing marginal convergence of MCMC chains and future work can help develop these tools.

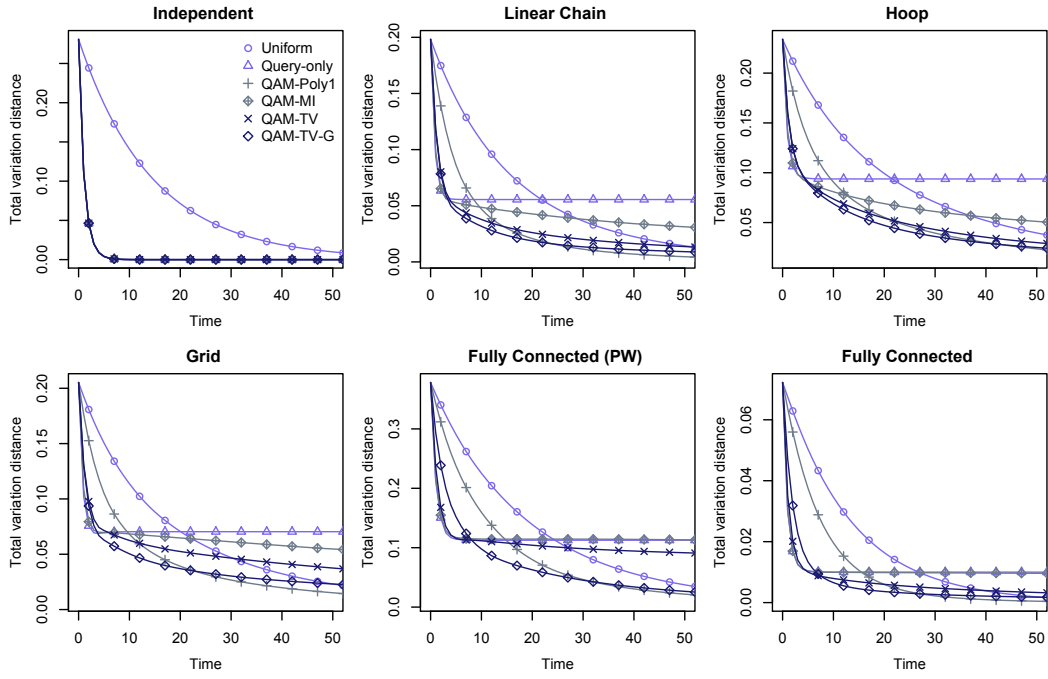

Figure 1: Convergence to the query marginals of the stationary distribution from an initial uniform distribution.

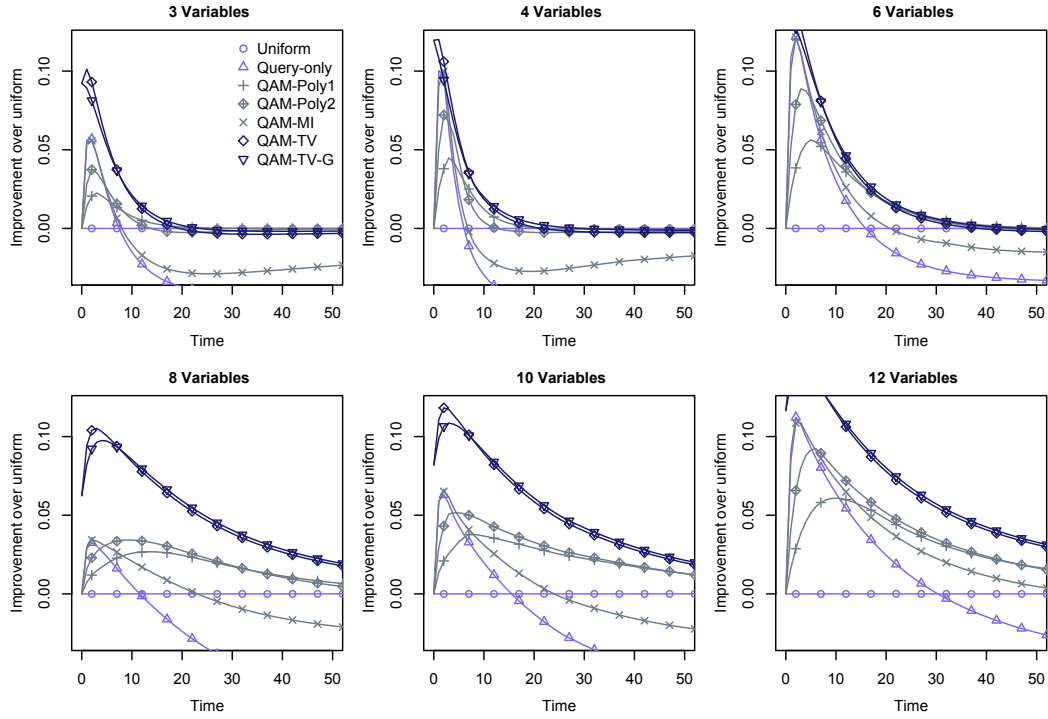

Figure 2: Improvement over uniform $p$ as the number of variables increases. Above the line $x = 0$ is an improvement in marginal convergence, and below is worse than the baseline. As number of variables increase, the improvements of the query specific techniques increase.

# 7 Acknowledgements

This work was supported in part by the Center for Intelligent Information Retrieval, in part by IARPA via DoI/NBC contract #D11PC20152, in part by IARPA and AFRL contract #FA8650-10-C-7060 , and in part by UPenn NSF medium IIS-0803847. The U.S. Government is authorized to reproduce and distribute reprint for Governmental purposes notwithstanding any copyright annotation thereon. Any opinions, findings and conclusions or recommendations expressed in this material are the authors' and do not necessarily reflect those of the sponsor. The authors would also like to thank Alexandre Passos and Benjamin Marlin for useful discussion.

## Footnotes

[1]Research in probabilistic databases has demonstrated that a large class of relational algebra queries can be represented as graphical models and answered using statistical queries of the this form [6, 7].

## References

[1] Yucheng Low, Joseph Gonzalez, Aapo Kyrola, Danny Bickson, Carlos Guestrin, and Joseph M. Hellerstein. Graphlab: A new parallel framework for machine learning. In *Conference on Uncertainty in Artificial Intelligence (UAI)*, Catalina Island, California, July 2010.

[2] Sameer Singh, Amarnag Subramanya, Fernando Pereira, and Andrew McCallum. Large-scale cross-document coreference using distributed inference and hierarchical models. In *Association for Computational Linguistics: Human Language Technologies (ACL HLT)*, 2011.

[3] Arthur Choi and Adnan Darwiche. Focusing generalizations of belief propagation on targeted queries. In *Association for the Advancement of Artificial Intelligence (AAAI)*, 2008.

[4] Anton Chechetka and Carlos Guestrin. Focused belief propagation for query-specific inference. In *International Conference on Artificial Intelligence and Statistics (AI STATS)*, 2010.

[5] Nilesh Dalvi and Dan Suciu. The dichotomy of conjunctive queries on probabilistic structures. Technical Report 0612102, University of Washington, 2007.

[6] Prithviraj Sen, Amol Deshpande, and Lise Getoor. Exploiting shared correlations in probabilistic databases. In *Very Large Data Bases (VLDB)*, 2008.

[7] Daisy Zhe Wang, Eirlinaios Michelakis, Minos Garofalakis, and Joseph M. Hellerstein. BayesStore: Managing large, uncertain data repositories with probabilistic graphical models. In *Very Large Data Bases (VLDB)*, 2008.

[8] Hoifung Poon and Pedro Domingos. Joint inference in information extraction. In *Association for the Advancement of Artificial Intelligence*, pages 913–918, Vancouver, Canada, 2007.

[9] Aron Culotta, Michael Wick, Robert Hall, and Andrew McCallum. First-order probabilistic models for coreference resolution. In *Human Language Technology Conf. of the North American Chapter of the Assoc. of Computational Linguistics (HLT/NAACL)*, pages 81–88, 2007.

[10] Adrian Barbu and Song Chun Zhu. Generalizing Swendsen-Wang to sampling arbitrary posterior probabilities. *IEEE Trans. Pattern Anal. Mach. Intell.*, 27(8):1239–1253, 2005.

[11] Ruslan Salakhutdinov and Geoffrey Hinton. Deep Boltzmann machines. In *International Conference on Artificial Intelligence and Statistics (AI STATS)*, 2009.

[12] Bhaskara Marthi, Hanna Pasula, Stuart Russell, and Yuval Peres. Decayed MCMC filtering. In *Conference on Uncertainty in Artificial Intelligence (UAI)*, pages 319–326, 2002.

[13] Michael Wick, Andrew McCallum, and Gerome Miklau. Scalable probabilistic databases with factor graphs and MCMC. In *Very Large Data Bases (VLDB)*, pages 794–804, 2010.

[14] Michael Wick, Andrew McCallum, and Gerome Miklau. Representing uncertainty in probabilistic databases with scalable factor graphs. Master's thesis, University of Massachusetts, proposed September 2008 and submitted April 2009.

[15] Daisy Zhe Wang, Michael J. Franklin, Minos Garofalakis, Joseph M. Hellerstein, and Michael L. Wick. Hybrid in-database inference for declarative information extraction. In *Proceedings of the 2011 international conference on Management of data*, SIGMOD '11, pages 517–528, New York, NY, USA, 2011. ACM.

[16] R.H. Swendsen and J.S. Wang. Nonuniversal critical dynamics in MC simulations. *Phys. Rev. Lett.*, 58(2):68–88, 1987.

[17] Radford Neal. Slice sampling. *Annals of Statistics*, 31:705–767, 2000.

